# Online Learning via Global Feedback for Phrase Recognition

**Xavier Carreras**      **Lluís Màrquez**
TALP Research Center, LSI Department
Technical University of Catalonia (UPC)
Campus Nord UPC, E–08034 Barcelona
{carreras,lluism}@lsi.upc.es

## Abstract

This work presents an architecture based on perceptrons to recognize phrase structures, and an online learning algorithm to train the perceptrons together and dependently. The recognition strategy applies learning in two layers: a filtering layer, which reduces the search space by identifying plausible phrase candidates, and a ranking layer, which recursively builds the optimal phrase structure. We provide a recognition-based feedback rule which reflects to each local function its committed errors from a global point of view, and allows to train them together online as perceptrons. Experimentation on a syntactic parsing problem, the recognition of clause hierarchies, improves state-of-the-art results and evinces the advantages of our global training method over optimizing each function locally and independently.

## 1   Introduction

Over the past few years, many machine learning methods have been successfully applied to Natural Language tasks in which phrases of some type have to be recognized. Generally, given an input sentence —as a sequence of words— the task is to predict a bracketing for the sentence representing a structure of phrases, either sequential or hierarchical. For instance, syntactic analysis of Natural Language provides several problems of this type, such as partial parsing tasks [1, 2], or even full parsing [3].

The general approach consists of decomposing the global phrase recognition problem into a number of local learnable subproblems, and infer the global solution from the outcomes of the local subproblems. For chunking problems —in which phrases are sequentially structured— the approach is typically to perform a tagging. In this case, local subproblems include learning whether a word *opens*, *closes*, or is *inside* a phrase of some type (noun phrase, verb phrase, . . . ), and the inference process consists of sequentially computing the optimal tag sequence which encodes the phrases, by means of dynamic programming [1, 4, 5]. When hierarchical structure has to be recognized, additional local decisions are required to determine the embedding of phrases, resulting in a more complex inference process which recursively builds the global solution [3, 2, 6, 7]. In general, a learning system for these tasks makes use of several learned functions which interact in some way to determine the structure.

A usual methodology for solving the local subproblems is to use a discriminative learning algorithm to learn a classifier for each local decision [1, 2]. Each individual classifier is trained separately from the others, maximizing some local measure such as the accuracy of the local decision. However, when performing the phrase recognition task, the classifiers are used together and dependently, in the sense that one classifier predictions' may affect the prediction of another. Indeed, the global performance of a system is measured in terms of precision and recall of the recognized phrases, which, although related, is not the local classification accuracy measure for which the local classifiers are usually trained.

In this direction, recent works in the area provide alternative strategies in which the learning process is driven from the global level. The general idea consists of moving the learning strategy from the binary classification setting to a general ranking context into which the global problem can be casted. Crammer and Singer [8] present a label-ranking algorithm, in which several perceptrons receive feedback from the ranking they produce over a training instance. Har-Peled et al. [9] study a general learning framework in which the constraints between a number of linear functions and an output prediction allow to effectively learn a desired label-ranking function. For structured outputs, and motivating this work, Collins [10] introduces a variant of the perceptron for tagging tasks, in which the learning feedback is globally given from the output of the Viterbi decoding algorithm.

In this paper we present a global learning strategy for the general task of recognizing phrases in a sentence. We adopt the general phrase recognition strategy of our previous work [6]. Given a sentence, learning is first applied at word level to identify phrase candidates of the solution. Then, learning is applied at a higher-order level in which phrase candidates are scored to discriminate among competing ones. The overall strategy infers the global solution by exploring with learning components a number of plausible solutions.

As a main contribution, we propose a recognition-based feedback rule which allows to learn the decisions in the system as perceptrons, all in one go. The learning strategy works online at sentence level. When visiting a sentence, the perceptrons are first used to recognize the set of phrases, and then updated according to the correctness of the global solution. As a result, each local function is automatically adapted to the recognition strategy. Furthermore, following [11] the final model incorporates *voted* prediction methods for the perceptrons and the use of kernel functions. Experimenting on the Clause Identification problem [2] we show the effectiveness of our method, evincing the benefits over local learning strategies and improving the best results for the particular task.

## 2  Phrase Recognition

### 2.1  Formalization

Let $x$ be a sentence formed by $n$ words $x_i$, with $i$ ranging from $0$ to $n-1$, belonging to the sentence space $\mathcal{X}$. Let $\mathcal{K}$ be a predefined set of phrase categories. For instance, in syntactic parsing $\mathcal{K}$ may include noun phrases, verb phrases, prepositional phrases and clauses, among others. A *phrase*, denoted as $(s,e)_k$, is the sequence of consecutive words spanning from word $x_s$ to word $x_e$, having $s \leq e$, with category $k \in \mathcal{K}$.

Let $ph_1 = (s_1, e_1)_{k_1}$ and $ph_2 = (s_2, e_2)_{k_2}$ be two different phrases. We define that $ph_1$ and $ph_2$ *overlap* iff $s_1 < s_2 \leq e_1 < e_2$ or $s_2 < s_1 \leq e_2 < e_1$, and we note it as $ph_1 \sim ph_2$. Also, we define that $ph_1$ is *embedded* in $ph_2$ iff $s_2 \leq s_1 \leq e_1 \leq e_2$, and we note it as $ph_1 \prec ph_2$.

Let $\mathcal{P}$ be the set of all possible phrases, expressed as $\mathcal{P} = \{(s,e)_k \mid 0 \leq s \leq e,\ k \in \mathcal{K}\}$. A *solution* for a phrase recognition problem is a set $y$ of phrases which is *coherent* with respect to some *constraints*. We consider two types of constraints: overlapping and embedding. For the problem of recognizing sequentially organized phrases, often referred to as *chunking*, phrases are not allowed to overlap or embed. Thus, the solution space can

be formally expressed as $\mathcal{Y} = \{y \subseteq \mathcal{P} \mid \forall ph_1, ph_2 \in y \;\; ph_1 \not\sim ph_2 \wedge ph_1 \not\prec ph_2\}$ . More generally, for the problem of recognizing phrases organized hierarchically, a solution is a set of phrases which do not overlap but may be embedded. Formally, the solution space is $\mathcal{Y} = \{y \subseteq \mathcal{P} \mid \forall ph_1, ph_2 \in y \;\; ph_1 \not\sim ph_2\}$ .

In order to evaluate a phrase recognition system we use the standard measures for recognition tasks: *precision* ($p$) —the ratio of recognized phrases that are correct—, *recall* ($r$) —the ratio of correct phrases that are recognized— and their harmonic mean $F_1 = \frac{2pr}{p+r}$.

## 2.2 Recognizing Phrases

The mechanism to recognize phrases is described here as a function which, given a sentence $x$, identifies the set of phrases $y$ of $x$: $\mathrm{R} : \mathcal{X} \to \mathcal{Y}$. We assume two components within this function, both being learning components of the recognizer. First, we assume a function $\mathrm{P}$ which, given a sentence $x$, identifies a set of candidate phrases, not necessarily coherent, for the sentence, $\mathrm{P}(x) \subseteq \mathcal{P}$. Second, we assume a score function which, given a phrase, produces a real-valued prediction indicating the plausability of the phrase for the sentence.

The phrase recognizer is a function which searches a coherent phrase set for a sentence $x$ according to the following optimality criterion:

$$\mathrm{R}(x) = \arg \max_{y \subseteq \mathrm{P}(x) \mid y \in \mathcal{Y}} \sum_{(s,e)_k \in y} \mathrm{score}((s,e)_k, x, y) \qquad (1)$$

That is, among all the coherent subsets of candidate phrases, the optimal solution is defined as the one whose phrases maximize the summation of phrase scores.

The function $\mathrm{P}$ is only used to reduce the search space of the $\mathrm{R}$ function. Note that the $\mathrm{R}$ function constructs the optimal phrase set by evaluating scores of phrase candidates, and, regarding the length of the sentence, there is a quadratic number of possible phrases, that is, the set $\mathcal{P}$. Thus, considering straightforwardly all phrases in $\mathcal{P}$ would result in a very expensive exploration. The function $\mathrm{P}$ is intended to filter out phrase candidates from $\mathcal{P}$ by applying decisions at word level. A simple setting for this function is a *start-end* classification for each phrase type: each word of the sentence is tested as *k-start* —if it is likely to start phrases of type $k$— and as *k-end* —if it is likely to end phrases type $k$. Each *k-start* word $x_s$ with each *k-end* word $x_e$, having $s \leq e$, forms the phrase candidates $(s, e)_k$. Assuming *start* and *end* binary classification functions, $\mathrm{h}_{\mathrm{S}}^k$ and $\mathrm{h}_{\mathrm{E}}^k$, for each type $k \in \mathcal{K}$, the filtering function is expressed as:

$$\mathrm{P}(x) = \{ (s, e)_k \in \mathcal{P} \mid \mathrm{h}_{\mathrm{S}}^k(x_s) = +1 \wedge \mathrm{h}_{\mathrm{E}}^k(x_e) = +1 \}$$

Alternatives to this setting may be to consider a single pair of *start-end* classifiers, independent of phrase types, or to perform a different tagging for identifying phrases, such as the well-known *begin-inside* classification. In general, each classifier will be applied to each word in the sentence, and deciding the best strategy for identifying phrase candidates will depend on the sparseness of phrases in a sentence, the length of phrases and the number of categories.

Once the phrase candidates are identified, the optimal coherent phrase set is selected according to (1). Due to its nature, there is no need to explicitly enumerate each possible coherent phrase set, which would result in an exponential exploration. Instead, by guiding the exploration through the problem constraints and using dynamic programming the optimal coherent phrase set can be found in polynomial time over the sentence length. For chunking problems, the solution can be found in quadratic time by performing a Viterbi-style exploration from left to right [4]. When embedding of phrases is allowed, a cubic-time bottom-up exploration is required [6]. As noted above, in either cases there will be the additional cost of applying a quadratic number of decisions for scoring phrases.

Summarizing, the phrase recognition system is performed in two layers: the identification layer, which filters out phrase candidates in linear time, and the scoring layer, which selects the optimal phrase chunking in quadratic or cubic time.

# 3 Additive Online Learning via Recognition Feedback

In this section we describe an online learning strategy for training the learning components of the Phrase Recognizer, namely the *start-end* classifiers in P and the *score* function. The learning challenge consists in approximating the functions so as to maximize the global $F_1$ measure on the problem, taking into account that the functions interact. In particular, the *start-end* functions define the actual input space of the *score* function.

Each function is implemented using a linear separator, $\mathrm{h_w} : \mathbb{R}^n \to \mathbb{R}$, operating in a feature space defined by a feature representation function, $\phi : \mathcal{X} \to \mathbb{R}^n$, for some instance space $\mathcal{X}$. The function P consists of two classifiers per phrase type: the *start* classifier ($\mathrm{h}_{\mathrm{S}}^k$) and the *end* classifier ($\mathrm{h}_{\mathrm{E}}^k$). Thus, the P function is formed by a prediction vector for each classifier, noted as $\mathbf{w}_{\mathrm{S}}^k$ or $\mathbf{w}_{\mathrm{E}}^k$, and a unique shared representation function $\phi_{\mathrm{w}}$ which maps a word in context into a feature vector. A prediction is computed as $\mathrm{h}_{\mathrm{S}}^k(x) = \mathbf{w}_{\mathrm{S}}^k \cdot \phi_{\mathrm{w}}(x)$, and similarly for the $\mathrm{h}_{\mathrm{E}}^k$, and the sign is taken as the binary classification. The *score* function computes a real-valued score for a phrase candidate $(s, e)_k$. We implement this function with a prediction vector $\mathbf{w}^k$ for each type $k \in \mathcal{K}$, and also a shared representation function $\phi_{\mathrm{p}}$ which maps a phrase into a feature vector. The score prediction is then given by the expression: $\mathrm{score}((s, e)_k, x, y) = \mathbf{w}^k \cdot \phi_{\mathrm{p}}((s, e)_k, x, y)$.

## 3.1 The FR-Perceptron Learning Algorithm

We propose a mistake-driven online learning algorithm for training the parameter vectors all together. We give the algorithm the name FR-Perceptron since it is a Perceptron-based learning algorithm that approximates the prediction vectors in P as Filters of words, and the score vectors as Rankers of phrases. The algorithm starts with all vectors initialized to $0$, and then runs repeatedly in a number of epochs $T$ through all the sentences in the training set. Given a sentence, it predicts its optimal phrase solution as specified in (1) using the current vectors. As in the traditional Perceptron algorithm, if the predicted phrase set is not perfect the vectors responsible of the incorrect prediction are updated additively. The algorithm is as follows:

- Input: $\{(x^1, y^1), \ldots, (x^m, y^m)\}$, $x^i$ are sentences, $y^i$ are solutions in $\mathcal{Y}$
- Define: $W = \{\mathbf{w}_{\mathrm{S}}^k, \mathbf{w}_{\mathrm{E}}^k, \mathbf{w}^k | k \in \mathcal{K}\}$.
- Initialize: $\forall \mathbf{w} \in W \; \mathbf{w} = 0$;
- for $t = 1 \ldots T$, for $i = 1 \ldots m$ :
    1. $\hat{y} = \mathrm{R}_W(x^i)$
    2. recognition_learning_feedback$(W, x^i, y^i, \hat{y})$
- Output: the vectors in $W$.

We now describe the recognition-based learning feedback. By analyzing the dependencies between each function and a solution, we derive a feedback rule which naturally fits the phrase recognition setting. Let $y^*$ be the gold set of phrases for a sentence $x$, and $\hat{y}$ the set predicted by the R function. Let $\mathrm{goldS}(x_i, k)$ and $\mathrm{goldE}(x_i, k)$ be, respectively, the perfect indicator functions for *start* and *end* boundaries of phrases of type $k$. That is, they return 1 if word $x_i$ starts/ends some $k$-phrase in $y^*$ and -1 otherwise. We differentiate three kinds of phrases in order to give feedback to the functions being learned:

- Phrases correctly identified: $\forall (s, e)_k \in y^* \cap \hat{y}$:
    - Do nothing, since they are correct.
- Missed phrases: $\forall (s, e)_k \in y^* \backslash \hat{y}$:
    1. Update misclassified boundary words:
       if $(\mathbf{w}_{\mathrm{S}}^k \cdot \phi_{\mathrm{w}}(x_s) \leq 0)$ then $\mathbf{w}_{\mathrm{S}}^k = \mathbf{w}_{\mathrm{S}}^k + \phi_{\mathrm{w}}(x_s)$
       if $(\mathbf{w}_{\mathrm{E}}^k \cdot \phi_{\mathrm{w}}(x_e) \leq 0)$ then $\mathbf{w}_{\mathrm{E}}^k = \mathbf{w}_{\mathrm{E}}^k + \phi_{\mathrm{w}}(x_e)$
    2. Update score function, if applied:
       if $(\mathbf{w}_{\mathrm{S}}^k \cdot \phi_{\mathrm{w}}(x_s) > 0 \wedge \mathbf{w}_{\mathrm{E}}^k \cdot \phi_{\mathrm{w}}(x_e) > 0)$ then $\mathbf{w}^k = \mathbf{w}^k + \phi_{\mathrm{p}}((s, e)_k, x, y)$
- Over-predicted phrases: $\forall (s, e)_k \in \hat{y} \backslash y^*$:
    1. Update score function:    $\mathbf{w}^k = \mathbf{w}^k - \phi_{\mathrm{p}}((s, e)_k, x, y)$
    2. Update words misclassified as S or E:
       if $(\mathrm{goldS}(x_s, k) = -1)$ then $\mathbf{w}_{\mathrm{S}}^k = \mathbf{w}_{\mathrm{S}}^k - \phi_{\mathrm{w}}(x_s)$
       if $(\mathrm{goldE}(x_e, k) = -1)$ then $\mathbf{w}_{\mathrm{E}}^k = \mathbf{w}_{\mathrm{E}}^k - \phi_{\mathrm{w}}(x_e)$

This feedback models the interaction between the two layers of the recognition process. The *start-end* layer filters out phrase candidates for the scoring layer. Thus, misclassifying the boundary words of a correct phrase blocks the generation of the candidate and produces a missed phrase. Therefore, we move the *start* or *end* prediction vectors toward the misclassified boundary words of a missed phrase. When an incorrect phrase is predicted, we move away the prediction vectors from the *start* or *end* words, provided that they are not boundary words of a phrase in the gold solution. Note that we deliberately do not care about false positives *start* or *end* words which do not finally over-produce a phrase.

Regarding the scoring layer, each category prediction vector is moved toward missed phrases and moved away from over-predicted phrases. It is important to note that this feedback operates only on the basis of the predicted solution $\hat{y}$, avoiding to make updates for every prediction the function has made. Thus, the learning strategy is taking advantage of the recognition process, and concentrates on (i) assigning high scores for the correct phrases and (ii) making the incorrect competing phrases to score lower than the correct ones. As a consequence, this feedback rule tends to approximate the desired behavior of the global R function, that is, to make the summation of the scores of the correct phrase set maximal with respect to other phrase set candidates. This learning strategy is closely related to other recent works on learning ranking functions [10, 8, 9].

**A Note on the Convergence**    Assuming linear separability for each *start*, *end* and *score* function, it can be shown that (i) the mistakes of the *start-end* filters are bounded (applying Novikoff's proof); (ii) between two consecutive updates in the *start-end* layer, there is room only for a finite number of updates of the *score* function; and (iii) once the *start-end* filters have converged, the correct solution is always considered in the *score* layer as candidate, and in this state the overall learning process converges (applying the proof of Collins for a perceptron tagger [10]).

## 4    Experiments on Clause Identification

Clause Identification is the problem of recognizing the clauses of a sentence. A clause can be roughly defined as a phrase with a subject, possibly implicit, and a predicate. Clauses in a sentence form a hierarchical structure which constitutes the skeleton of the full syntactic tree. In the following example, the clauses are annotated with brackets:
> *( ( (When (you don't have any other option)), it is easy (to fight) .)*

We followed the setting of the CoNLL-2001 competition [1]. The problem consists of recognizing the set of clauses on the basis of words, part-of-speech tags (PoS), and syntactic base phrases (or chunks). There is only one category of phrases to be considered, namely the clauses. The data consists of a training set (8,936 sentences, 24,841 clauses), a development set (2,012 sentences, 5,418 clauses) and a test set (1,671 sentences, 5,225 clauses).

**Representation Functions**   We now describe the representation functions $\phi_\mathrm{w}$ and $\phi_\mathrm{p}$, which respectively map a word or a phrase and their local context into a feature vector in $\{0, 1\}^n$. Their design is inspired in our previous work [6]. For the function $\phi_\mathrm{w}(x_i)$ we capture the form, PoS and chunk tags of words in a *window* around $x_i$, that is, words $x_{i+l}$ with $l \in [-L_\mathrm{w}, +L_\mathrm{w}]$. Each attribute type, together with each relative position $l$ and each returned value forms a final binary indicator feature (for instance, "the word at position -2 is that" is a binary feature). Also, we consider the word decisions of the words to the left of $x_i$, that is, binary flags indicating whether the $[-L_\mathrm{w}, -1]$ words in the window are starts and/or ends of a phrase. For the function $\phi_\mathrm{p}(s, e)$ we represent the context of the phrase by capturing a $[-L_\mathrm{p}, 0]$ window of forms, PoS and chunks at the $s$ word, and a separate $[0, +L_\mathrm{p}]$ window at the $e$ word. Furthermore, we represent the $(s, e)$ phrase by evaluating a pattern from $s$ to $e$ which captures the relevant elements in the sentence fragment from word $s$ to word $e$ [2]. We experimentally set both $L_\mathrm{w}$ and $L_\mathrm{p}$ to 3.

On this problem we were interested in comparing the FR-Perceptron algorithm versus other alternative learning methods. The system to train was composed by the *start* and *end* functions which identify clause candidates, and a score function for clauses. As alternatives, we first considered a batch classification setting, in which each function is trained separately with binary classification loss. To do so, we generated three data sets from training examples, one for each function. For the *start-end* sets, we considered an example for each word in the data. To train the *score* classifier, we generated only the phrase candidates formed with all pairs of correct phrase boundaries. This latter generation greatly reduces the real instance space in which the scoring function operates. The alternative of generating all possible phrases as examples would be more realistic, but infeasible for the learning algorithm since it would produce 1,377,843 examples, with a 98.2% of negatives. As a secondary intermediate approach, we considered a simple model which learns all the functions online via binary classification loss. That is, the training sentences are visited online as in the FR-Perceptron: first, the *start-end* functions are applied to each word, and according to their positive decisions, phrase examples are generated to train the *score* function. In this way, the input of the *score* function is dynamically adapted to the *start-end* behavior, but a classification feedback is given to each function for each decision taken.

The functions of the system were actually modeled as *Voted Perceptrons* [11], which compute a prediction as an *average* of all vectors generated during training. For the batch classification setting, we modeled the functions as Voted Perceptrons and also as SVMs[3]. In all cases, a function can be expressed in dual form as a combination of training instances, which allows the use of kernel functions. We work with polynomial kernels of degree 2. [4]

We trained the perceptron models for up to 20 epochs via the FR-Perceptron algorithm and via classification feedback, either online (CO-VP) or batch (CB-VP). We also trained SVM classifiers (Cl-SVM), adjusting the soft margin $C$ parameter on the development set.

Figure 1: Performance on the development set with respect to the number of epochs. Top: global $F_1$ (left) and precision/recall on *starts* (right). Bottom: given the *start-end* filters, upper bound on the global $F_1$ (left) and number of proposed phrase candidates (right).

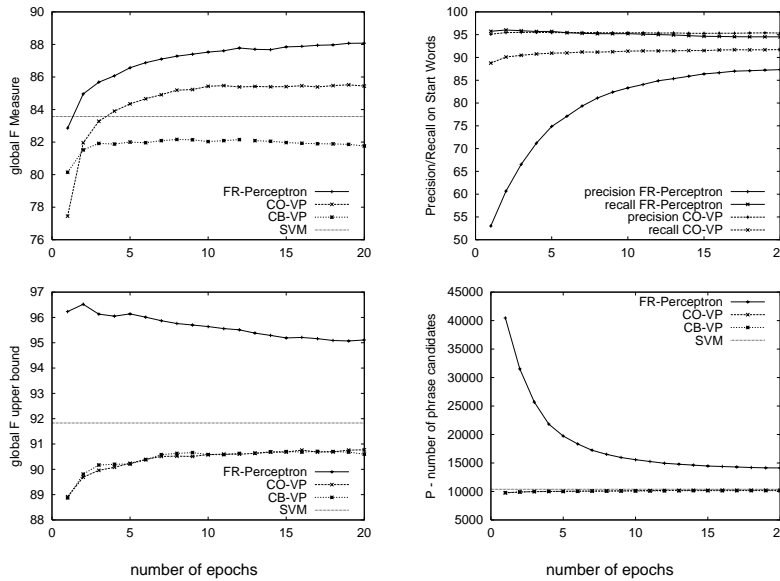

Figure 1 (top, left) shows the performance curves in terms of the $F_1$ measure with respect to the number of training epochs. Clearly, the FR-Perceptron model exhibits a much better curve than classification models, being at any epoch more than 2 points higher than the online model, and far from the batch models. To get an idea of how the learning strategy behaves, it is interesting to look at the other plots of Figure 1. The top right plot shows the performance of the *start* function. The FR-Perceptron model exhibits the desirable filtering behavior for this local decision, which consists in maintaining a very high recall (so that no correct candidates are blocked) while increasing the precision during epochs. In contrast, the CO-VP model concentrates mainly on the precision. The same behavior is observed for the other classification models, and also for the *end* local decision. The *start-end* behavior is also shown from a global point of view at the bottom plots. The left plot shows the maximum achievable global $F_1$, assuming a perfect scorer, given the phrases proposed by the *start-end* functions. Additionally, the right plot depicts the filtering capabilities in terms of the number of phrase candidates produced, out of a total number of 300,511 possible phrases. The FR-Perceptron behavior in the filtering layer is clear: while it maintains a high recall on identifying correct phrases (above 95%), it substantially reduces the number of phrase candidates to explore in the scoring layer, and thus, it progressively simplifies the input to the *score* function. Far from this behavior, the classification-based models are not sensitive to the global performance in the filtering layer and, although they aggressively reduce the search space, provide only a moderate upper bound on the global $F_1$.

Table 4 shows the performance of each model, together with the results of our previous system [6], which held the best results on the problem. There, the same decisions were learned by AdaBoost classifiers working in a richer feature space. Also, the score function was a robust combination of several classifiers. These were trained taking into account the errors of the start-end classifiers, which required a tuning procedure to select the amount of introduced errors. Our new approach is much simpler to learn, since the interaction between functions is naturally ruled by the recognition feedback. Looking at results, we substantially improve the global $F_1$.

|  | T | development | | | test | | |
|---|---|---|---|---|---|---|---|
|  |  | prec. | recall | $F_{\beta=1}$ | prec. | recall | $F_{\beta=1}$ |
| CB-VP | 8 | 83.84 | 80.55 | 82.16 | 82.22 | 78.09 | 80.10 |
| SVM | - | 84.31 | 82.83 | 83.57 | 83.19 | 80.00 | 81.57 |
| CO-VP | 19 | 91.06 | 80.62 | 85.52 | 89.25 | 77.62 | 83.03 |
| FR-Perceptron | 20 | 90.56 | **85.73** | **88.08** | 88.17 | **82.10** | **85.03** |
| AdaBoost [6] | – | **92.53** | 82.48 | 87.22 | **90.18** | 78.11 | 83.71 |

Table 1: Results of Clause Identification on the CoNLL-2001 development and test sets. The T column shows the optimal number of epochs on the development set.

## 5   Conclusion

We have presented a global learning strategy for the general problem of recognizing structures of phrases, in which, typically, several different learning functions interact to explore and recognize the structure. The effectiveness of our method has been empirically proved in the problem of clause identification, where we have shown that a considerable improvement can be obtained by exploiting high-order global dependencies in learning, in contrast to concentrating only on the local subproblems. These results suggest to scale up global learning strategies to more complex problems found in the natural language area (such as full parsing or machine translation), or other structured domains.

### Acknowledgements

Research partially funded by the European Commission (Meaning, IST-2001-34460) and the Spanish Research Department (Hermes, TIC2000-0335-C03-02; Petra, TIC2000-1735-C02-02). Xavier Carreras is supported by a grant from the Catalan Research Department.

## Footnotes

[1]Data and details at the CoNLL-2001 website: *http://cnts.uia.ac.be/conll2001* .

[2]The following elements are considered in a pattern: a) Punctuation marks and coordinate conjunctions; b) The word *that*; c) Relative pronouns; d) Verb phrase chunks; and e) The top clauses within the $s$ to $e$ fragment, already recognized through the bottom up search (a clause in a pattern reduces all the elements within it into an atomic element).

[3]We used the SVM$^{light}$ package available at *http://svmlight.joachims.org* .

[4]Initial tests revealed poor performance for the linear case and no improvements for degrees $> 2$.

## References

[1] E. F. Tjong Kim Sang and S. Buchholz. Introduction to the CoNLL-2000 Shared Task: Chunking. In *Proc. of CoNLL-2000 and LLL-2000*, 2000.

[2] Erik F. Tjong Kim Sang and Hervé Déjean. Introduction to the CoNLL-2001 Shared Task: Clause Identification. In *Proc. of CoNLL-2001*, 2001.

[3] A. Ratnaparkhi. Learning to Parse Natural Language with Maximum-Entropy Models. *Machine Learning*, 34(1):151–175, 1999.

[4] V. Punyakanok and D. Roth. The Use of Classifiers in Sequential Inference. In *Advances in Neural Information Processing Systems 13 (NIPS'00)*, 2001.

[5] T. Kudo and Y. Matsumoto. Chunking with Support Vector Machines . In *Proc. of 2nd Conference of the North American Chapter of the Association for Computational Linguistics*, 2001.

[6] X. Carreras, L. Màrquez, V. Punyakanok, and D. Roth. Learning and Inference for Clause Identification. In *Proceedings of the 14th ECML*, Helsinki, Finland, 2002.

[7] T. Kudo and Y. Matsumoto. Japanese Dependency Analyisis using Cascaded Chunking . In *Proc. of CoNLL-2002*, 2002.

[8] K. Crammer and Y. Singer. A Family of Additive Online Algorithms for Category Ranking. *Journal of Machine Learning Research*, 3:1025–1058, 2003.

[9] S. Har-Peled, D. Roth, and D. Zimak. Constraint Classification for Multiclass Classification and Ranking. In *Advances in Neural Information Processing Systems 15 (NIPS'02)*, 2003.

[10] M. Collins. Discriminative Training Methods for Hidden Markov Models: Theory and Experiments with Perceptron Algorithms. In *Proceedings of the EMNLP'02*, 2002.

[11] Y. Freund and R. E. Schapire. Large Margin Classification Using the Perceptron Algorithm. *Machine Learning*, 37(3):277–296, 1999.
